# Speech Denoising and Dereverberation Using Probabilistic Models

**Hagai Attias**  **John C. Platt**  **Alex Acero**  **Li Deng**
Microsoft Research
1 Microsoft Way
Redmond, WA 98052
{*hagaia,jplatt,alexac,deng*}@*microsoft.com*

## Abstract

This paper presents a unified probabilistic framework for denoising and dereverberation of speech signals. The framework transforms the denoising and dereverberation problems into Bayes-optimal signal estimation. The key idea is to use a strong speech model that is pre-trained on a large data set of clean speech. Computational efficiency is achieved by using variational EM, working in the frequency domain, and employing conjugate priors. The framework covers both single and multiple microphones. We apply this approach to noisy reverberant speech signals and get results substantially better than standard methods.

## 1 Introduction

This paper presents a statistical-model-based algorithm for reconstructing a speech source from microphone signals recorded in a stationary noisy reverberant environment. Speech enhancement in a realistic environment is a challenging problem, which remains largely unsolved in spite of more than three decades of research. Speech enhancement has many applications and is particularly useful for robust speech recognition [7] and for telecommunication.

The difficulty of speech enhancement depends strongly on environmental conditions. If a speaker is close to a microphone, reverberation effects are minimal and traditional methods can handle typical moderate noise levels. However, if the speaker is far away from a microphone, there are more severe distortions, including large amounts of noise and noticeable reverberation. Denoising and dereverberation of speech in this condition has proven to be a very difficult problem [4].

Current speech enhancement methods can be placed into two categories: single-microphone methods and multiple-microphone methods. A large body of literature exists on single-microphone speech enhancement methods. These methods often use a probabilistic framework with statistical models of a single speech signal corrupted by Gaussian noise [6, 8]. These models have not been extended to dereverberation or multiple microphones.

Multiple-microphone methods start with microphone array processing, where an array of microphones with a known geometry is deployed to make both spatial and temporal measurements of sounds. A microphone array offers significant advantages compared to single microphone methods. Non-adaptive algorithms can denoise a signal reasonably well, as

long as it originates from a limited range of azimuth. These algorithms do not handle reverberation, however. Adaptive algorithms can handle reverberation to some extent [4], but existing methods are not derived from a principled probabilistic framework and hence may be sub-optimal.

Work on blind source separation has attempted to remove the need for fixed array geometries and pre-specified room models. Blind separation attempts the full multi-source, multi-microphone case. In practice, the most successful algorithms concentrate on instantaneous noise-free mixing with the same number of sources as sensors and with very weak probabilistic models for the source [5]. Some algorithms for noisy non-square instantaneous mixing have been developed [1], as well as algorithms for convolutive square noise-free, mixing [9]. However, the full problem including noise and convolution has so far remained open.

In this paper, we present a new method for speech denoising and dereverberation. We use the framework of probabilistic models, which allows us to integrate the different aspects of the whole problem, including strong speech models, environmental noise and reverberation, and microphone arrays. This integration is performed in a principled manner facilitating a coherent unified treatment. The framework allows us to produce a Bayes-optimal estimation algorithm. Using a strong speech model leads to computational intractability, which we overcome using a variational approach. The computational efficiency is further enhanced by working in the frequency domain and by employing conjugate priors. The resulting algorithm has complexity $O(N \log N)$. Results on noisy speech show significant improvement over standard methods.

Due to space limitations, the full derivation and mathematical details for this method are provided in the technical report [3].

**Notation and conventions**. We work with time series data using a frame-by-frame analysis with $N$-point frames. Thus, all signals and systems, e.g. $y_n^i$, have a time point subscript extending over $n = 0, ..., N-1$. With the superscript $i$ omitted, $y_n$ denotes all microphone signals. When $n$ is also omitted, $y$ denotes all signals at all time points. Superscripts may become subscripts and vice versa when no confusion arises. The discrete Fourier transform (DFT) of $x_n$ is $\tilde{x}_k = \sum_n \exp(-i\omega_k n)x_n$. We define the primed quantity

$$\tilde{a}_k' = 1 - \sum_{n=1}^{p} e^{-i\omega_k n} a_n \tag{1}$$

for variables $a_n$ with $n = 1, ..., p$.

The Gaussian distribution for a random vector $a$ with mean $\mu$ and precision matrix $V$ (defined as the inverse covariance matrix) is denoted $\mathcal{N}(a \mid \mu, V)$. The Gamma distribution for a non-negative random variable $\nu$ with $\alpha$ degrees of freedom and inverse scale $\beta$ is denoted $\mathcal{G}(\nu \mid \alpha, \beta) \propto \nu^{\alpha/2-1} \exp(-\beta\nu/2)$. Their product, the Normal-Gamma distribution

$$\mathcal{NG}(a, \nu \mid \mu, V, \alpha, \beta) = \mathcal{N}(a \mid \mu, \nu V)\mathcal{G}(\nu \mid \alpha, \beta) , \tag{2}$$

turns out to be particularly useful. Notice that it relates the precision of $a$ to $\nu$.

**Problem Formulation** We consider the case where a single speech source is present and $M$ microphones are available. The treatment of the single-microphone case is a special case of $M = 1$, but is not qualitatively different.

Let $x_n$ be the signal emitted by the source at time $n$, and let $y_n^i$ be the signal received at microphone $i$ at the same time. Then

$$y_n^i = h_n^i \star x_n + u_n^i = \sum_m h_m^i x_{n-m} + u_n^i , \tag{3}$$

where $h_m^i$ is the impulse response of the filter (of length $K^i \leq N$) operating on the source as it propagates toward microphone $i$, $\star$ is the convolution operator, and $u_n^i$ denotes the

noise recorded at that microphone. Noise may originate from both microphone responses and from environmental sources.

In a given environment, the task is to provide an optimal estimate of the clean speech signal $x$ from the noisy microphone signals $y^i$. This requires the estimation of the convolving filters $h^i$ and characteristics of the noise $u^i$. This estimation is accomplished by Bayesian inference on probabilistic models for $x$ and $u^i$.

## 2 Probabilistic Signal Models

We now turn to our model for the speech source. Much of the work on speech denoising in the past has usually employed very simple source models: AR or ARMA descriptions [6]. One exception is [8], which uses an HMM whose observations are Gaussian AR models. These simple denoising models incorporate very little information on the structure of speech. Such an approach *a priori* allows any value for the model coefficients, including values that are unlikely to occur in a speech signal. Without a strong prior, it is difficult to estimate the convolving filters accurately due to identifiability. A source prior is especially important in the single microphone case, which estimates $N$ clean samples plus model coefficients from $N$ noisy samples. Thus, the absence of a strong speech model degrades reconstruction quality.

The most detailed statistical speech models available are those employed by state-of-the-art speech recognition engines. These systems are generally based on mixture of diagonal Gaussian models in the mel-cepstral domain. These models are endowed with temporal Markov dynamics and have a very large ($\approx$ 100000) number of states corresponding to individual atoms of speech. However, in the mel-cepstral domain, the noisy reverberant speech has a strong non-linear relationship to the clean speech.

**Physical speech production model**. In this paper, we work in the linear time/frequency domain using a statistical model and take an intermediate approach regarding the model size. We model speech production with an AR($p$) model:

$$x_n = \sum_{m=1}^{p} a_m x_{n-m} + v_n \,, \tag{4}$$

where the coefficients $a_m$ are related to the physical shape of a "lossless tube" model of the vocal tract.

To turn this physical model into a probabilistic model, we assume that $v_n$ are independent zero-mean Gaussian variables with scalar precision $\nu$. Each speech frame $x = (x_0, ..., x_{N-1})$ has its own parameters $\theta = (a_1, ..., a_p, \nu)$. Given $\theta$, the joint distribution of $x$ is generally a zero-mean Gaussian, $p(x \mid \theta) = \mathcal{N}(x \mid 0, \Lambda)$, where $\Lambda$ is the $N \times N$ precision matrix. Specifically, the joint distribution is given by the product

$$p(x \mid \theta) = \prod_n \mathcal{N}(x_n \mid \sum_m a_m x_{n-m}, \nu). \tag{5}$$

**Probabilistic model in the frequency domain**. However, rather than employing this product form directly, we work in the frequency domain and use the DFT to write

$$p(x \mid \theta) \propto \exp(-\frac{\nu}{2N} \sum_{k=0}^{N-1} \mid \tilde{a}'_k \mid^2 \mid \tilde{x}_k \mid^2) \,, \tag{6}$$

where $\tilde{a}'_k$ is defined in (1). The precision matrix $\Lambda$ is now given by an inverse DFT, $\Lambda_{nm} = (\nu/N) \sum_k e^{i\omega_k(n-m)} \mid \tilde{a}'_k \mid^2$. This matrix belongs to a sub-class of Toeplitz matrices called *circulant Toeplitz*. It follows from (6) that the mean power spectrum of $x$ is related to $\theta$ via $S_k = \langle \mid \tilde{x}_k \mid^2 \rangle = N/(\nu \mid \tilde{a}'_k \mid^2)$.

**Conjugate priors**. To complete our speech model, we must specify a distribution over the speech production parameters $\theta$. We use a $S$-state mixture model with a Normal-Gamma distribution (2) for each component $s = 1, ..., S$: $p(\theta \mid s) = \mathcal{N}(a_1, ..., a_p \mid \mu_s, \nu V_s)\mathcal{G}(\nu \mid \alpha_s, \beta_s)$. This form is chosen by invoking the idea of a *conjugate prior*, which is defined as follows. Given the model $p(x \mid \theta)p(\theta \mid s)$, the prior $p(\theta \mid s)$ is conjugate to $p(x \mid \theta)$ iff the *posterior* $p(\theta \mid x, s)$, computed by Bayes' rule, has the same functional form as the prior. This choice has the advantage of being quite general while keeping the clean speech model analytically tractable.

It turns out, as discussed below, that significant computational savings result if we restrict the $p \times p$ precision matrices $V_s$ to have a circulant Toeplitz structure. To do this without having to impose an explicit constraint, we reparametrize $p(\theta \mid s)$ in terms of $\xi_n^s, \eta_n^s$ instead of $\mu_n^s, V_{nm}^s$, and work in the frequency domain:

$$p(\theta \mid s) \propto \exp(-\frac{\nu}{2p} \sum_{k=0}^{p-1} \mid \tilde{\xi}_k^s \tilde{a}_k - \tilde{\eta}_k^s \mid^2) \cdot \nu^{-\frac{\alpha_s}{2}} \exp(-\frac{\beta_s}{2}\nu) . \tag{7}$$

Note that we use a $p$- rather than $N$-point DFT. The precisions are now given by the inverse DFT $V_{nm}^s = (1/p) \sum_k e^{i\omega_k(n-m)} \mid \tilde{\xi}_k^s \mid^2$ and are manifestly circulant. It is easy to show that conjugacy still holds.

Finally, the mixing fractions are given by $p(s) = \pi_s$. This completes the specification of our clean speech model $p(x)$ in terms of the latent variable model $p(x, \theta, s) = p(x \mid \theta)p(\theta \mid s)p(s)$. The model is parametrized by $W = (\xi_m^s, \eta_m^s, \alpha_s, \beta_s, \pi_s)$.

**Speech model training**. We pre-train the speech model parameters $W$ using 10000 sentences of the Wall Street Journal corpus, recorded with a close-talking microphone for 150 male and female speakers of North American English. We used 16msec overlapping frames with $N = 256$ time points at 16kHz sampling rate. Training was performed using an EM algorithm derived specifically for this model [3]. We used $S = 256$ clusters and $p = 12$. $W$ were initialized by extracting the AR($p$) coefficients from each frame using the autocorrelation method. These coeffcients were converted into cepstral coefficients, and clustered into $S$ classes by $k$-means clustering. We then considered the corresponding hard clusters of the AR($p$) coefficients, and separately fit a model $p(\theta \mid s)$ (7) to each. The resulting parameters were used as initial values for the full EM algorithm.

**Noise model**. In this paper, we use an AR($q$) description for the noise recorded by microphone $i$, $u_n^i = \sum_m b_m^i u_{n-m}^i + w_n^i$. The noise parameters are $\phi^i = (b_m^i, \lambda^i)$, where $\lambda^i$ are the precisions of the zero-mean Gaussian excitations $w_n^i$. In the frequency domain we have the joint distribution

$$p(u^i \mid \phi^i) \propto \exp(-\frac{\lambda^i}{2N} \sum_{k=0}^{N-1} \mid \tilde{b}_{i,k}' \mid^2 \mid \tilde{u}_k^i \mid^2) . \tag{8}$$

As in (6), the parameters $\phi^i$ determine the spectra of the noise. But unlike the speech model, the AR($q$) noise model is chosen for mathematical convenience rather than for its relation to an underlying physical model.

**Noisy speech model**. The form (8) now implies that given the clean speech $x$, the distribution of the data $y^i$ is

$$p(y^i \mid x) \propto \exp(-\frac{\lambda^i}{2N} \sum_{k=0}^{N-1} \mid \tilde{b}_{i,k}' \mid^2 \mid \tilde{y}_k^i - \tilde{h}_k^i \tilde{x}_k \mid^2) . \tag{9}$$

This completes the specification of our noisy speech model $p(y)$ in terms of the joint distribution $\prod_i p(y^i \mid x)p(x \mid \theta)p(\theta \mid s)p(s)$.

## 3   Variational Speech Enhancement (VSE) Algorithm

The denoising and dereverberation task is accomplished by estimating the clean speech $x$, which requires estimating the speech parameters $\theta$, the filter coefficients $h^i$, and the noise parameters $\phi^i$. These tasks can be performed by the EM algorithm. This algorithm receives the data $y^i$ from an utterance (a long sequence of frames) as input and proceeds iteratively. In the E-step, the algorithm computes the sufficient statistics of the clean speech $x$ and the production parameters $\theta$ for each frame. In the M-step, the algorithm uses the sufficient statistics to update the values of $h^i$ and $\phi^i$, which are assumed unchanged throughout the utterance. This assumption limits the current VSE algorithm to stationary noise and reverberation. Source reconstruction is performed as a by-product of the E-step.

**Intractability and variational EM.** In the clean speech model $p(x)$ above, inference (i.e. computing $p(s, \theta \mid x)$ for the observed clean speech $x$) is tractable. However, in the noisy case, $x$ is hidden and consequently inference becomes intractable. The posterior $p(s, \theta, x \mid y)$ includes a quartic term $\exp(x^2 \theta^2)$, originating from the product of two Gaussian variables, which causes the intractability.

To overcome this problem, we employ a variational approach [10]. We replace the exact posterior distribution over the hidden variables by an approximate one, $q(s, \theta, x \mid y)$, and select the optimal $q$ by maximizing

$$\mathcal{F}[q] = \sum_s \int dx \, d\theta \; q(s, \theta, x \mid y) \log \frac{p(s, \theta, x, y)}{q(s, \theta, x \mid y)} \tag{10}$$

w.r.t. $q$. To achieve tractability, we must restrict the space of possible $q$. We use the partially factorized form

$$q = q(s) q(\theta \mid s) q(x \mid s) , \tag{11}$$

where the $y$-dependence of $q$ is omitted. Given $y$, this distribution defines a mixture model for $x$ and a mixture model for $\theta$, while maintaining correlations between $x$ and $\theta$ (i.e., $q(x, \theta) \neq q(x) q(\theta)$). Maximizing $\mathcal{F}$ is equivalent to minimizing the KL distance between $q$ and the exact conditional $p(s, \theta, x \mid y)$ under the restriction (11).

With no further restriction, the functional form of $q$ falls out of free-form optimization, as shown in [2]. For the production parameters, $q(\theta \mid s)$ turns out to have the form $q(\theta \mid s) = \mathcal{N}(a_1, ..., a_p \mid \hat{\mu}_s, \nu \hat{V}_s) \mathcal{G}(\nu \mid \hat{\alpha}_s, \hat{\beta}_s)$. This form is functionally identical to that of the prior $p(\theta \mid s)$, consistent with the conjugate prior idea. The parameters of $q$ are distinguished from the prior's by the $\hat{\phantom{x}}$ symbol. Similarly, the state responsibilities are $q(s) = \hat{\pi}_s$. For the clean speech, we obtain Gaussians, $q(x \mid s) = \mathcal{N}(x \mid \rho_s, \Lambda_s)$, with state-dependent means and precisions.

**E-step and Wiener filtering.** To derive the E-step, we first ignore reverberation by setting $h_n^i = \delta_{n,0}$ and assuming a single microphone signal $y_n$, thus omitting $i$. The extension to multiple microphones and reverberation is straightforward.

The parameters of $q$ are estimated at the E-step from the noisy speech in each frame, using an iterative algorithm. First, the parameters of $q(\theta \mid s)$ are updated via

$$\hat{V}_s = R_s + V_s , \qquad \hat{\mu}_s = \hat{V}_s^{-1}(r_s + V_s \mu_s) , \tag{12}$$

where $R_{nm}^s = (1/N) \sum_k e^{i\omega_k(n-m)} E_s(\mid \tilde{x}_k \mid^2)$, $r_n^s = R_{n0}^s$, and $E_s$ denotes averaging w.r.t. $q(x \mid s)$, which is easily done analytically. The update rules for $\hat{\alpha}_s, \hat{\beta}_s, \hat{\pi}_s$ are shown in [3].

Next, the parameters of $q(x \mid s)$ are obtained by inverse DFT via

$$\rho_n^s = \frac{1}{N} \sum_{k=0}^{N-1} e^{i\omega_k n} \tilde{f}_k^s \, \tilde{y}_k , \qquad \Lambda_{nm}^s = \frac{1}{N} \sum_{k=0}^{N-1} e^{i\omega_k(n-m)} \tilde{g}_k^s , \tag{13}$$

where $\tilde{f}_k^s = \lambda \mid \tilde{b}_k' \mid^2 /\tilde{g}_k^s$, and $\tilde{g}_k^s = \lambda \mid \tilde{b}_k' \mid^2 + E_s(\nu \mid \tilde{a}_k' \mid^2)$. Here $E_s$ denotes averaging w.r.t. $q(\theta \mid s)$. These steps are iterated to convergence, upon which the estimated speech signal for this frame is given by the weighted sum $\hat{x} = \sum_s \hat{\pi}_s \rho_s$.

We point out that the correspondence between AR parameters and spectra implies the Wiener filter form $\tilde{f}_k^s = S_k^s/(S_k^s + N_k)$, where $S_k^s$ is the estimated clean speech spectrum associated with state $s$, and $N_k$ is the noise spectrum, both at frequency $\omega_k$. Hence, the updated $\rho_s$ in (13) is obtained via a state-dependent Wiener filter, and the clean speech is estimated by a sum of Wiener filters weighted by the state responsibilities. The same Wiener structure holds in the presence of reverberation. Notice that, whereas the conventional Wiener filter is linear and obtained directly from the known speech spectrum, our filters depend nonlinearly on the data, since the unknown speech spectra and state responsibilities are estimated iteratively by the above algorithm.

**M-step.** After computing the sufficient statistics of $\theta, x$ for each frame, $\phi^i$ and $h^i$ are updated using the whole utterance. The update rules are shown in [3]. Alternatively, the $\phi^i$ can be estimated directly by maximum likelihood if a non-speech portion of the input signal can be found.

**Computational savings.** The complexity of the updates for $q(x \mid s)$ and $q(\theta \mid s)$ is $N \log N$ and $Sp \log p$, respectively. This is due to working in the frequency domain, using the FFT algorithm to perform the DFT, and by using conjugate priors and circulant precisions. Working in the time domain and using priors with general precisions would result in the considerably higher complexity of $N^2$ and $Sp^3$, respectively.

## 4   Experiments

**Denoising.** We tested this algorithm on 150 speech sentences by male and female speakers from the Wall Street Journal (WSJ) database, which were not included in the training set. These sentences were distorted by adding either synthetic noise (white or pink), or noise recorded in an office environment with a PC and air conditioning. The distortions were applied at different SNRs. All of these noises were stationary. We then applied the algorithm to estimate the noise parameters and reconstruct the original speech signal. The result was compared with a sophisticated, subband-based implementation of the spectral subtraction (SS) technique.

**Denoising & Dereverberation.** We tested this algorithm on 100 WSJ sentences, which were distorted by convolving them with a 10-tap artificial filter and adding synthetic white Gaussian noise at different SNRs. We then applied the algorithm to estimate both the noise level and the filter. Here we used a simpler speech model with $p(\theta \mid s) = \delta(\theta - \theta_s)$.

**Speech Recognition.** We also examined the potential contribution of this algorithm to robust speech recognition, by feeding the denoised signals as inputs to a recognition system. The system used a version of the Microsoft continuous-density HMM (Whisper), with 6000 tied HMM states (senones), 20 Gaussians per state, and the speech represented via Mel-cepstrum, delta cepstrum, and delta-delta cepstrum. A fixed bigram language model is used in all the experiments. The system had been trained on a total of $16,000$ female clean speech sentences. The test set consisted of 167 female WSJ sentences, which were distorted by adding synthetic white non-Gaussian noise. The word error rate was 55.06% under the training-test mismatched condition of no preprocessing on the test set and decoding by HMMs trained with clean speech. This condition is the baseline for the relative performance improvement listed in the last row of Table 1. For these experiments, we compared VSE to the SS algorithm described in [7].

Table 1 shows that the Variational Speech Enhancement (VSE) algorithm is superior to SS at removing stationary noise either measured via SNR improvement or via relative reduction in speech recognition error rate (compared to baseline).

| | dB noise added | reverb added | SS synthetic noise | SS real noise | VSE synthetic noise | VSE real noise |
|---|---|---|---|---|---|---|
| SNR improvement | 5 | No | 4.3 | 4.3 | 6.0 | 5.5 |
| SNR improvement | 10 | No | 4.1 | 4.1 | 5.8 | 5.1 |
| SNR improvement | 5 | Yes | 6.7 | | 10.2 | |
| SNR improvement | 10 | Yes | 8.3 | | 13.2 | |
| Speech recognition *relative* improvement | 10 | No | 38.6% | | 65.1% | |

Table 1: Experimental Results.

## 5  Conclusion

We have presented a probabilistic framework for denoising and dereverberation. The framework uses a strong speech model to perform Bayes-optimal signal estimation. The parameter estimation and the reconstruction of the signal are performed using a variational EM algorithm. Working in the frequency domain and using conjugate priors leads to great computational savings. The framework applies equally well to one-microphone and multiple-microphone cases. Experiments show that the optimal estimation can outperform standard methods such as spectral subtraction. Future directions include adding temporal dynamics to the speech model via an HMM structure, using a richer adaptive noise model (e.g. a mixture), and handling non-stationary noise and filters.

## References

[1] H. Attias. Independent factor analysis. *Neural Computation*, 11(4):803–851, 1999.

[2] H. Attias. A variational bayesian framework for graphical models. In T. Leen, editor, *Advances in Neural Information Processing Systems*, volume 12, pages 209–215. MIT Press, 2000.

[3] H. Attias, J. C. Platt, A. Acero, and L. Deng. Speech denoising and dereverberation using probabilistic models: Mathematical details. Technical Report MSR-TR-2001-02, Microsoft Research, 2001. http://research.microsoft.com/~hagaia.

[4] M. S. Brandstein. On the use of explicit speech modeling in microphone array applications. In *Proc. ICASSP*, pages 3613–3616, 1998.

[5] J.-F. Cardoso. Infomax and maximum likelihood for source separation. *IEEE Signal Processing Letters*, 4(4):112–114, 1997.

[6] A. Dembo and O. Zeitouni. Maximum a posteriori estimation of time-varying ARMA processes from noisy observations. *IEEE Trans. Acoustics, Speech, and Signal Processing*, 36(4):471–476, 1988.

[7] L. Deng, A. Acero, M. Plumpe, and X. D. Huang. Large-vocabulary speech recognition under adverse acoustic environments. In *Proceedings of the International Conference on Spoken Language Processing*, volume 3, pages 806–809, 2000.

[8] Y. Ephraim. Statistical-model-based speech enhancement systems. *Proc. IEEE*, 80(10):1526–1555, 1992.

[9] J. C. Platt and F. Faggin. Networks for the separation of sources that are superimposed and delayed. In J. E. Moody, editor, *Advances in Neural Information Processing Systems*, volume 4, pages 730–737, 1992.

[10] L. K. Saul, T. Jaakkola, and M. I. Jordan. Mean field theory of sigmoid belief networks. *J. Artificial Intelligence Research*, 4:61–76, 1996.
